# Model Based Image Compression and Adaptive Data Representation by Interacting Filter Banks

**Toshiaki Okamoto, Mitsuo Kawato, Toshio Inui**
ATR Auditory and
Visual Perception Research Laboratories
Sanpeidani, Inuidani, Seika-cho, Soraku-gun
Kyoto 619-02, Japan

**Sei Miyake**
NHK Science and
Technical Research Laboratories
1-10-11, Kinuta, Setagaya
Tokyo 157, Japan

## Abstract

To achieve high-rate image data compression while maintainig a high quality reconstructed image, a good image model and an efficient way to represent the specific data of each image must be introduced. Based on the physiological knowledge of multi-channel characteristics and inhibitory interactions between them in the human visual system, a mathematically coherent parallel architecture for image data compression which utilizes the Markov random field image model and interactions between a vast number of filter banks, is proposed.

## 1. Introduction

Data compression has been one of the most important and active areas in information theory and computer science. The goal of image coding is reducing the number of bits in data representation as much as possible, and reconstructing a faithful duplicate of the original image. In order to achieve a high compression ratio while maintaining the high quality

of the reconstructed image, a good image model and an efficient way to represent image data must be found. Based on physiological knowledge of the human visual system, we propose a mathematically coherent parallel architecture for the image data compression, which utilizes a stochastic image model and interactions between a vast number of filter banks.

## 2. Model based image compression and dynamic spatial filtering

The process of reconstructing an original image from compressed data is an ill-posed problem, since an infinite number of original images lead to the same compressed data and solutions to the inverse problem can not uniquely be determined. The coupled Markov random field (MRF) image model proposed by Geman and Geman is introduced to resolve this ill-posedness. The mean field approximation of the MRF is equivalent to a recurrent type neural network with the Ljapunov function (see Koch. Marroquin and Yuille as a special case where the form of the Ljapunov function is predetermined). Correspondingly, a similar deterministic framework of image compression in which the MRF is replaced by the recurrent network, can be developed.

Further, even if a good MRF model is introduced for a family of images, the data for each image must be known in order to reconstruct it. In previous studies of image data compression, representation of image data is fixed in each schema. On the other hand, in this paper, an adaptive data representation is proposed, tuned to each image by competion and cooperation of a vast number of filter banks.

Fig. 1 shows a block diagram of the proposed communication system. Procedures at the encoder side are (1) partial partition and segmentation of the image by the

line process of the MRF which represents the image discontinuity, (2) learning of energy parameters which uses the line process to define the MRF model in each segmented area of the image, (3) adaptive data representation of images by cooperation and competition of a vast number of filter banks. (4) Information about energy value parameters, the types of selected filter and their outputs, and the line processes is transmitted, through communication channel. (5) Image reconstruction is carried out at the decorder site by stochastic relaxation based on the aquired MRF model, output from the selected filters, and the line process. These procedures are explained in detail below.

1. The set of line processes represents discontinuities in the 3-dimensional world such as occluding contours or boundaries between different objects. It is not necessarily closed, but it can posess a strong tendency to do so if the MRF model is appropriately chosen. Based on this property, the image can be partially segmented into several regions.

2. If we adopt the MRF image model, the occurrence probability $\Pi(\omega)$ of each configuration $\omega$ is Gibbsian :

$$\Pi(\omega)=\frac{\exp\{-U(\omega)/T\}}{Z}$$

Furthermore, the energy $U(\omega)$ can be expressed as a summation of local potential $V_c(\omega)$, which depends on the configuration only in the clique $C$.

$$U(\omega)=\sum_{C\in S_c} V_c(\omega)$$

Determination of the local energy $V_c$ is equivalent to defining a specific MRF model of the image. Determination of the local energy is equivalent to assigning a real value $V_{\ell}$, to

every possible configuration within the clique $C$. These energy parameters are estimated so that the Kullback divergence $G$ between the real image distribution $P$ and the model image distribution $P'$ is minimized :

$$G(V) = \sum_{\omega} P(\omega) \log\{\frac{P(\omega)}{P'(\omega, V_c)}\}$$

The following learning equation can be derived in approximately the same way as the learning rule of the Boltzmann machine (Ackley, Hinton, Sejnowski, 1985).

$$\Delta V_{\xi_i} = -\varepsilon \frac{\partial G}{\partial V_{\xi_i}} = -\eta\{<\sum_{C \in S_c} I_i(C)>_{real} - <\sum_{C \in S_c} I_i(C)>\}>_{desir}$$

Here $I_i(C)$ is the characteristic function of the specific configuration $\xi_i$ of the clique $C$, that is, $I_i(C)=1$ if $\{y_s; s \in C\} = \xi_i$ otherwise, $I_i(C)=0$. The first term on the right side is the average number of configurations in the real image. The second term on the right side is the average number of each configuration $\xi_i$ generated in the MRF with the energy $V_c$ when part of the image configuration is fixed to the given image.

3. This procedure is based on the multi-channel characteristics of the human visual system, inhibitory interaction between X-cell and Y-cell systems, and interactions between columns with different orientation selectivity, etc. We prepare a vast number of filters centered at each site $s$ in a variety of sizes, shapes and orientations. In particular, we use two-dimensional Gaussian filters $G_s(\omega)$ to represent the DC components (i.e. average luminance) of the gray level, and use the first-order derivative of the Gaussian filters $\nabla G_s(\omega)$ to represent the gradient of the gray levels. The filters whose receptive fields significantly intersect with the line process are inhibited. Inhibitory interactions between filters of similar, shape and orientation at nearby sites are introduced

as well as self excitation to find the N-maximum outputs of $\nabla G_s$, and to find the N-minimum outputs of the Laplacian Gaussian $\Delta G_s$. Of course, 2N must be less than the number of sites to attain data compression.

4. We transmit the local potential energy, the site of the line process, and the outputs from the N − maximum, and the outputs from the N Gaussain filters which correspond to the N − minimum Laplacian Gaussain filters.

5. Image reconstruction is carried out by the usual stochastic relaxation, that is, energy minimization with simulated annealing. However, because we have data constraints as output from the 2N selected filters, we need to minimize the sum of the MRF model energy and the data constraint energy :

$$E_{post}(\omega)=U(\omega)+\lambda\{\sum_{k=1}^{N}(\tilde{G_k}-G_k(\omega))^2+\sum_{k=1}^{N}(\nabla\tilde{G_k}-\nabla G_k(\omega))^2\}$$

If we do not further compress the filter outputs, the regularization parameter is increased to infinity during constrained stochastic relaxation.

## 3. Experimental results

First, we ascertained that the proposed energy learning rule works well for various images. Here, we report only one example from the data compression experiments. We used the shown in Fig. 2a to examine the potential of our scheme. The image data consists of 256 pixels, each of which has 8 bit gray levels. We used the dynamic spatial sampling of filter banks. Fig 2a also shows selected sample points in the image as black dots, as well as a few examples of selected filter shapes. Note that not only the density of the sampling points, but also the selected filter shapes are

appropriate local characteristics of the image. Fig. 2b shows the reconstructed image after 20 iterations of the relaxation computation. The signal to noise ratio of the reconstructed images was about 38dB.

## References

D. H., Ackley, G. E. Hinton, and T. J. Sejnowski, : "A Learning algorithm for Boltzmann Machines.", Cognitive Science, vol. 9, pp.147 − 169, (1985).

S. Geman and D. Geman, : "Stochastic relaxation, Gibbs distribution, and the Basian restoration of images", IEEE Trans. vol. PAMI − 6, pp.721 − 741, (1984).

S. Hongo, M. Kawato, T. Inui, and S. Miyake, ; "Contour extraction of images on parallel computer", Proc. of 1th IJCNN, (1989).

T. Inui, M. Kawato and R. Suzuki : "The mechanism of mental scanning in foveal vision", Biol. Cybern. vol. 30, pp.147 − 155, (1978).

C. Koch, J. Marroquin, and A. Yuille : "Analog 'neural' networks in early vision", Proc. Natl. Acad. Sci. USA, vol. 83, pp.4263 − 4267, (1986).

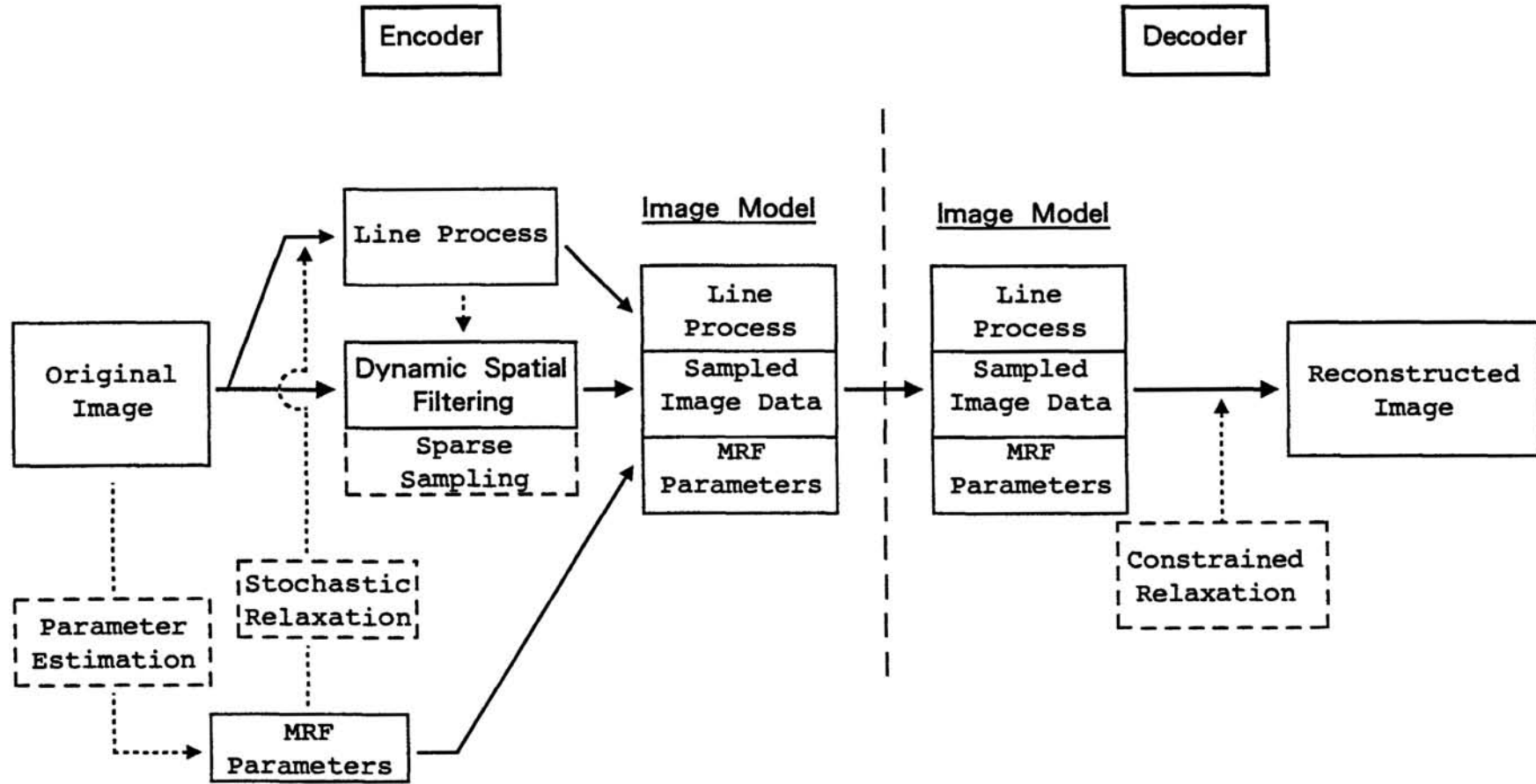

Fig. 1 Model Based Communication System

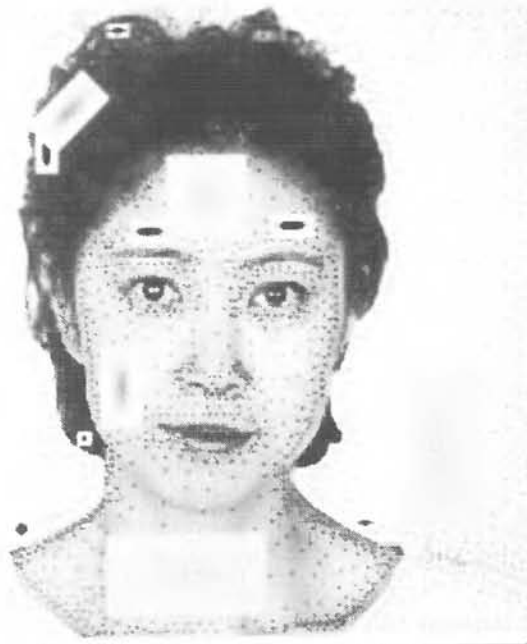

(a)   sampled  data  points
and  filters

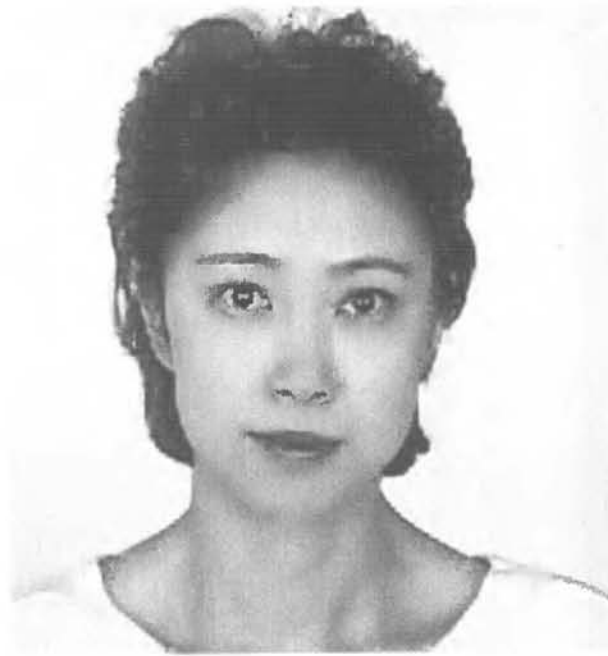

(b)   reconstructed  image

# Fig. 2 Computer  simulation
# of  image  data  compression
